# Forward-backward retraining of recurrent neural networks

Andrew Senior *          Tony Robinson
Cambridge University Engineering Department
Trumpington Street, Cambridge, England

## Abstract

This paper describes the training of a recurrent neural network as the letter posterior probability estimator for a hidden Markov model, off-line handwriting recognition system. The network estimates posterior distributions for each of a series of frames representing sections of a handwritten word. The supervised training algorithm, backpropagation through time, requires target outputs to be provided for each frame. Three methods for deriving these targets are presented. A novel method based upon the forward-backward algorithm is found to result in the recognizer with the lowest error rate.

## 1 Introduction

In the field of off-line handwriting recognition, the goal is to read a handwritten document and produce a machine transcription. Such a system could be used for a variety of purposes, from cheque processing and postal sorting to personal correspondence reading for the blind or historical document reading. In a previous publication (Senior 1994) we have described a system based on a recurrent neural network (Robinson 1994) which can transcribe a handwritten document.

The recurrent neural network is used to estimate posterior probabilities for character classes, given frames of data which represent the handwritten word. These probabilities are combined in a hidden Markov model framework, using the Viterbi algorithm to find the most probable state sequence.

To train the network, a series of targets must be given. This paper describes three methods that have been used to derive these probabilities. The first is a naive bootstrap method, allocating equal lengths to all characters, used to start the training procedure. The second is a simple Viterbi-style segmentation method that assigns a single class label to each of the frames of data. Such a scheme has been used before in speech recognition using recurrent networks (Robinson 1994). This representation, is found to inadequately represent some frames which can represent two letters, or the ligatures between letters. Thus, by analogy with the forward-backward algorithm (Rabiner and Juang 1986) for HMM speech recognizers, we have developed a

forward-backward method for retraining the recurrent neural network. This assigns a probability distribution across the output classes for each frame of training data, and training on these 'soft labels' results in improved performance of the recognition system.

This paper is organized in four sections. The following section outlines the system in which the neural network is used, then section 3 describes the recurrent network in more detail. Section 4 explains the different methods of target estimation and presents the results of experiments before conclusions are presented in the final section.

## 2   System background

The recurrent network is the central part of the handwriting recognition system. The other parts are summarized here and described in more detail in another publication (Senior 1994). The first stage of processing converts the raw data into an invariant representation used as an input to the neural network. The network outputs are used to calculate word probabilities in a hidden Markov model.

First, the scanned page image is automatically segmented into words and then normalized. Normalization removes variations in the word appearance that do not affect its identity, such as rotation, scale, slant, slope and stroke thickness. The height of the letters forming the words is estimated, and magnifications, shear and thinning transforms are applied, resulting in a more robust representation of the word. The normalized word is represented in a compact canonical form encoding both the shape and salient features. All those features falling within a narrow vertical strip across the word are termed a frame. The representation derived consists of around 80 values for each of the frames, denoted $x_t$. The $\tau$ frames $(x_1, \ldots, x_\tau)$ for a whole word are written $x_1^\tau$. Five frames would typically be enough to represent a single character. The recurrent network takes these frames sequentially and estimates the posterior character probability distribution given the data: $P(\Lambda_i | x_1^t)$, for each of the letters, a,..,z, denoted $\Lambda_0, \ldots, \Lambda_{25}$. These posterior probabilities are scaled by the prior class probabilities, and are treated as the emission probabilities in a hidden Markov model.

A separate model is created for each word in the vocabulary, with one state per letter. Transitions are allowed only from a state to itself or to the next letter in the word. The set of states in the models is denoted $Q = \{q_1, \ldots, q_N\}$ and the letter represented by $q_i$ is given by $L(q_i)$, $L : Q \mapsto \Lambda_0, \ldots, \Lambda_{25}$.

Word error rates are presented for experiments on a single-writer task tested with a 1330 word vocabulary[1]. Statistical significance of the results is evaluated using Student's $t$-test, comparing word recognition rates taken from a number of networks trained under the same conditions but with different random initializations. The results of the $t$-test are written: $T$(degrees of freedom) and the tabulated values: $t_{\text{significance}}$(degrees of freedom).

## 3   Recurrent networks

This section describes the recurrent error propagation network which has been used as the probability distribution estimator for the handwriting recognition system. Recurrent networks have been successfully applied to speech recognition (Robinson 1994) but have not previously been used for handwriting recognition, on-line or off-line. Here a left-to-right scanning process is adopted to map the frames of a word into a sequence, so adjacent frames are considered in consecutive instants.

A recurrent network is well suited to the recognition of patterns occurring in a time-series because series of arbitrary length can be processed, with the same processing being performed on each section of the input stream. Thus a letter '*a*' can be recognized by the same process, wherever it occurs in a word. In addition, internal 'state' units are available to encode multi-frame context information so letters spread over several frames can be recognized. The recurrent network

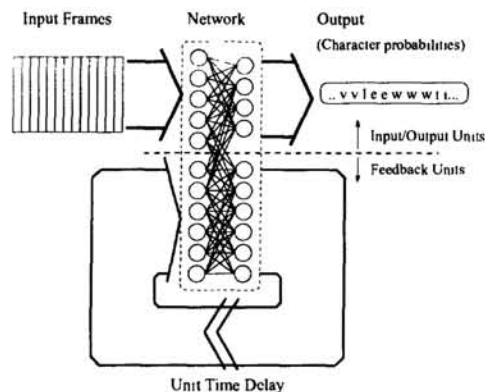

Figure 1: A schematic of the recurrent error propagation network. For clarity only a few of the units and links are shown.

architecture used here is a single layer of standard perceptrons with nonlinear activation functions. The output $o_i$ of a unit $i$ is a function of the inputs $a_j$ and the network parameters, which are the weights of the links $w_{ij}$ with a bias $b_i$:

$$o_i \;=\; f_i(\{\sigma_j\}), \qquad (1) \qquad \sigma_i \;=\; b_i + \sum a_k w_{ik}. \qquad (2)$$

The network is fully connected — that is, each input is connected to every output. However, some of the input units receive no external input and are connected one-to-one to corresponding output units through a unit time-delay (figure 1). The remaining input units accept a single frame of parametrized input and the remaining 26 output units estimate letter probabilities for the 26 character classes. The feedback units have a standard sigmoid activation function (3), but the character outputs have a 'softmax' activation function (4).

$$f_i(\{\sigma_j\}) \;=\; (1+e^{-\sigma_i})^{-1} \qquad (3) \qquad f_i(\{\sigma_j\}) \;=\; \frac{e^{\sigma_i}}{\sum_j e^{\sigma_j}}. \qquad (4)$$

During recognition ('forward propagation'), the first frame is presented at the input and the feedback units are initialized to activations of 0.5. The outputs are calculated (1 and 2) and read off for use in the Markov model. In the next iteration, the outputs of the feedback units are copied to the feedback inputs, and the next frame presented to the inputs. Outputs are again calculated, and the cycle is repeated for each frame of input, with a probability distribution being generated for each frame.

To allow the network to assimilate context information, several frames of data are passed through the network before the probabilities for the first frame are read off, previous output probabilities being discarded. This input/output latency is maintained throughout the input sequence, with extra, empty frames of inputs being presented at the end to give probability distributions for the last frames of true inputs. A latency of two frames has been found to be most satisfactory in experiments to date.

## 3.1  Training

To be able to train the network the target values $\zeta_j(t)$ desired for the outputs $o_j(x_t)$ $j = 0, \ldots, 25$ for frame $x_t$ must be specified. The target specification is dealt

with in the next section. It is the discrepancy between the actual outputs and these targets which make up the objective function to be maximized by adjusting the internal weights of the network. The usual objective function is the mean squared error, but here the relative entropy, $G$, of the target and output distributions is used:

$$G = -\sum_t \sum_j \zeta_j(t) \log \frac{\zeta_j(t)}{o_j(x_t)}. \tag{5}$$

At the end of a word, the errors between the network's outputs and the targets are propagated back using the generalized delta rule (Rumelhart *et al.* 1986) and changes to the network weights are calculated. The network at successive time steps is treated as adjacent layers of a multi-layer network. This process is generally known as 'back-propagation through time' (Werbos 1990). After processing $\tau$ frames of data with an input/output latency, the network is equivalent to a $(\tau +$ latency) layer perceptron sharing weights between layers. For a detailed description of the training procedure, the reader is referred elsewhere (Rumelhart *et al.* 1986; Robinson 1994).

## 4   Target re-estimation

The data used for training are only labelled by word. That is, each image represents a single word, whose identity is known, but the frames representing that word are not labelled to indicate which part of the word they represent. To train the network, a label for each frame's identity must be provided. Labels are indicated by the state $S_t \in Q$ and the corresponding letter $L(S_t)$ of which a frame $x_t$ is part.

### 4.1   A simple solution

To bootstrap the network, a naive method was used, which simply divided the word up into sections of equal length, one for each letter in the word. Thus, for an $N$-letter word of $\tau$ frames, $x_1^\tau$, the first letter was assumed to be represented by frames $x_1^{\frac{\tau}{N}}$, the next by $x_{\frac{\tau}{N}+1}^{\frac{2\tau}{N}}$ and so on. The segmentation is mapped into a set of targets as follows:

$$\zeta_j(t) = \begin{cases} 1 \text{ if } L(S_t) = \Lambda_j \\ 0 \text{ otherwise.} \end{cases} \tag{6}$$

Figure 2a shows such a segmentation for a single word. Each line, representing $\zeta_j(t)$ for some $j$, has a broad peak for the frames representing letter $\Lambda_j$. Such a segmentation is inaccurate, but can be improved by adding prior knowledge. It is clear that some letters are generally longer than others, and some shorter. By weighting letters according to their *a priori* lengths it is possible to give a better, but still very simple, segmentation. The letters '*i, l*' are given a length of $\frac{1}{2}$ and '*m, w*' a length $\frac{3}{2}$ relative to other letters. Thus in the word '*wig*', the first half of the frames would be assigned the label 'w', the next sixth 'i' and the last third the label 'g'. While this segmentation is constructed with no regard for the data being segmented, it is found to provide a good initial approximation from which it is possible to train the network to recognize words, albeit with high error rates.

### 4.2   Viterbi re-estimation

Having trained the network to some accuracy, it can be used to calculate a good estimate of the probability of each frame belonging to any letter. The probability of any state sequence can then be calculated in the hidden Markov model, and the most likely state sequence through the correct word $S^*$ found using dynamic programming. This best state sequence $S^*$ represents a new segmentation giving a label for each frame. For a network which models the probability distributions well, this segmentation will be better than the automatic segmentation of section 4.1

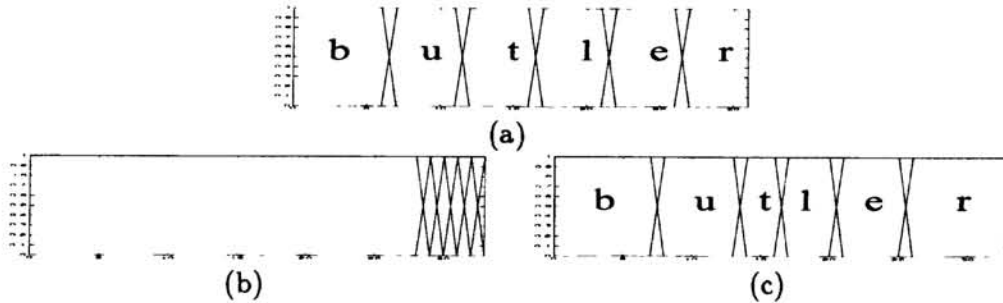

Figure 2: Segmentations of the word '*butler*'. Each line represents $P(S_t = \Lambda_i | S)$ for one letter $\Lambda_i$ and is high for frame $t$ when $S_t^* = \Lambda_i$. (a) is the equal-length segmentation discussed in section 4.1 (b) is a segmentation of an untrained network. (c) is the segmentation re-estimated with a trained network.

since it takes the data into account. Finding the most probable state sequence $S^*$ is termed a *forced alignment*. Since only the correct word model need be considered, such an alignment is faster than the search through the whole lexicon that is required for recognition. Training on this automatic segmentation gives a better recognition rate, but still avoids the necessity of manually segmenting any of the database.

Figure 2 shows two Viterbi segmentations of the word '*butler*'. First, figure 2b shows the segmentation arrived at by taking the most likely state sequence before training the network. Since the emission probability distributions are random, there is nothing to distinguish between the state sequences, except slight variations due to initial asymmetry in the network, so a poor segmentation results. After training the network (2c), the durations deviate from the prior assumed durations to match the observed data. This re-estimated segmentation represents the data more accurately, so gives better targets towards which to train. A further improvement in recognition accuracy can be obtained by using the targets determined by the re-estimated segmentation. This cycle can be repeated until the segmentations do not change and performance ceases to improve. For speed, the network is not trained to convergence at each iteration.

It can be shown (Santini and Del Bimbo 1995) that, assuming that the network has enough parameters, the network outputs after convergence will approximate the posterior probabilities $P(\Lambda_i | x_1^t)$. Further, the approximation $P(\Lambda_i | x_1^t) \approx P(\Lambda_i | x_t)$ is made. The posteriors are scaled by the class priors $P(\Lambda_i)$ (Bourlard and Morgan 1993), and these scaled posteriors are used in the hidden Markov model in place of data likelihoods since, by Bayes' rule,

$$P(x_t | \Lambda_i) \quad \propto \quad \frac{P(\Lambda_i | x_t)}{P(\Lambda_i)}. \tag{7}$$

Table 1 shows word recognition error rates for three 80-unit networks trained towards fixed targets estimated by another network, and then retrained, re-estimating the targets at each iteration. The retraining improves the recognition performance $(T(2) = 3.91, t_{.95}(2) = 2.92)$.

### 4.3  Forward-backward re-estimation

The system described above performs well and is the method used in previous recurrent network systems, but examining the speech recognition literature, a potential method of improvement can be seen. Viterbi frame alignment has so far been used to determine targets for training. This assigns one class to each frame, based on the most likely state sequence. A better approach might be to allow a distribution across all the classes indicating which are likely and which are not, avoiding a

Table 1: Error rates for 3 networks with 80 units trained with fixed
alignments, and retrained with re-estimated alignments.

| Training | Error (%) | |
|---|---|---|
| method | $\hat{\mu}$ | $\hat{\sigma}$ |
| Fixed targets | 21.2 | 1.73 |
| Retraining | 17.0 | 0.68 |

'hard' classification at points where a frame may indeed represent more than one
class (such as where slanting characters overlap), or none (as in a ligature). A 'soft'
classification would give a more accurate portrayal of the frame identities.

Such a distribution, $\gamma_p(t) = P(S_t = q_p|x_1^\tau, W)$, can be calculated with the *forward-
backward* algorithm (Rabiner and Juang 1986). To obtain $\gamma_p(t)$, the forward prob-
abilities $\alpha_p(t) = P(S_t = q_p, x_1^t)$ must be combined with the *backward* probabilities
$\beta_p(t) = P(S_t = q_p, x_{t+1}^\tau)$. The forward and backward probabilities are calculated
recursively in the same manner.

$$\alpha_r(t+1) = \sum_p \alpha_p(t)P(x_t|L(q_p))a_{p,r}, \tag{8}$$

$$\beta_p(t-1) = \sum_r a_{p,r}P(x_t|S_t = q_r)\beta_r(t). \tag{9}$$

Suitable initial distributions $\alpha_r(0) = \pi_r$ and $\beta_r(\tau+1) = \rho_r$ are chosen, *e.g.* $\pi$ and
$\rho$ are one for respectively the first and last character in the word, and zero for the
others. The likelihood of observing the data $x_1^\tau$ and being in state $q_p$ at time $t$ is
then given by:

$$\xi_p(t) = \alpha_p(t)\beta_p(t). \tag{10}$$

Then the probabilities $\gamma_p(t)$ of being in state $q_p$ at time $t$ are obtained by normal-
ization and used as the targets $\zeta_j(t)$ for the recurrent network character probability
outputs:

$$\gamma_p(t) = \frac{\xi_p(t)}{\sum_r \xi_r(t)}. \quad (11) \qquad \zeta_j(t) = \sum_{p:L(q_p)=\Lambda_j} \gamma_p(t). \quad (12)$$

Figure 3a shows the initial estimate of the class probabilities for a sample of the
word '*butler*'. The probabilities shown are those estimated by the forward-backward
algorithm when using an untrained network, for which the $P(x_t|S_t = q_p)$ will be
independent of class. Despite the lack of information, the probability distributions
can be seen to take reasonable shapes. The first frame must belong to the first
letter, and the last frame must belong to the last letter, of course, but it can also
be seen that half way through the word, the most likely letters are those in the
middle of the word. Several class probabilities are non-zero at a time, reflecting
the uncertainty caused since the network is untrained. Nevertheless, this limited
information is enough to train a recurrent network, because as the network begins
to approximate these probabilities, the segmentations become more definite. In
contrast, using Viterbi segmentations from an untrained network, the most likely
alignment can be very different from the true alignment (figure 2b). The segmen-
tation is very definite though, and the network is trained towards the incorrect
targets, reinforcing its error. Finally, a trained network gives a much more rigid
segmentation (figure 3b), with most of the probabilities being zero or one, but with
a boundary of uncertainty at the transitions between letters. This uncertainty,
where a frame might truly represent parts of two letters, or a ligature between
two, represents the data better. Just as with Viterbi training, the segmentations
can be re-estimated after training and retraining results in improved performance.
The final probabilistic segmentation can be stored with the data and used when
subsequent networks are trained on the same data. Training is then significantly
quicker than when training towards the approximate bootstrap segmentations and
re-estimating the targets.

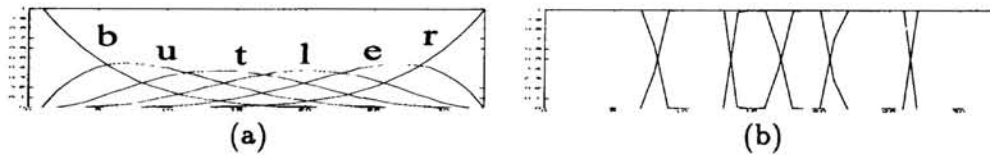

Figure 3: Forward-backward segmentations of the word 'butler'. (a) is the segmentation of an untrained network with a uniform class prior. (b) shows the segmentation after training.

The better models obtained with the forward-backward algorithm give improved recognition results over a network trained with Viterbi alignments. The improvement is shown in table 2. It can be seen that the error rates for the networks trained with forward-backward targets are lower than those trained on Viterbi targets ($T(2) = 5.24, t_{.975}(2) = 4.30$).

Table 2: Error rates for networks with 80 units trained with Viterbi or Forward-Backward alignments.

| Training method | Error (%) | |
|---|---|---|
| | $\hat{\mu}$ | $\hat{\sigma}$ |
| Viterbi | 17.0 | 0.68 |
| Forward-Backward | 15.4 | 0.74 |

## 5 Conclusions

This paper has reviewed the training methods used for a recurrent network, applied to the problem of off-line handwriting recognition. Three methods of deriving target probabilities for the network have been described, and experiments conducted using all three. The third method is that of the forward-backward procedure, which has not previously been applied to recurrent neural network training. This method is found to improve the performance of the network, leading to reduced word error rates. Other improvements not detailed here (including duration models and stochastic language modelling) allow the error rate for this task to be brought below 10%.

**Acknowledgments**
The authors would like to thank Mike Hochberg for assistance in preparing this paper.

## Footnotes

*Now at IBM T.J.Watson Research Center, Yorktown Heights NY10598, USA.

[1]The experimental data are available in ftp://svr-ftp.eng.cam.ac.uk/pub/data

## References

BOURLARD, H. and MORGAN, N. (1993) *Connectionist Speech Recognition: A Hybrid Approach*. Kluwer.

RABINER, L. R. and JUANG, B. H. (1986) An introduction to hidden Markov models. *IEEE ASSP magazine* **3** (1): 4–16.

ROBINSON, A. (1994) The application of recurrent nets to phone probability estimation. *IEEE Transactions on Neural Networks*.

RUMELHART, D. E., HINTON, G. E. and WILLIAMS, R. J. (1986) Learning internal representations by error propagation. In *Parallel Distributed Processing: Explorations in the Microstructure of Cognition*, ed. by D. E. Rumelhart and J. L. McClelland, volume 1, chapter 8, pp. 318–362. Bradford Books.

SANTINI, S. and DEL BIMBO, A. (1995) Recurrent neural networks can be trained to be maximum a posteriori probability classifiers. *Neural Networks* **8** (1): 25–29.

SENIOR, A. W., (1994) *Off-line Cursive Handwriting Recognition using Recurrent Neural Networks*. Cambridge University Engineering Department Ph.D. thesis. URL: ftp://svr-ftp.eng.cam.ac.uk/pub/reports/senior_thesis.ps.gz.

WERBOS, P. J. (1990) Backpropagation through time: What it does and how to do it. *Proceedings of the IEEE* **78**: 1550–60.